# Regression with Input-dependent Noise: A Gaussian Process Treatment

**Paul W. Goldberg**
Department of Computer Science
University of Warwick
Coventry, CV4 7AL, UK
pwg@dcs.warwick.ac.uk

**Christopher K.I. Williams**
Neural Computing Research Group
Aston University
Birmingham B4 7ET, UK
c.k.i.williams@aston.ac.uk

**Christopher M. Bishop**
Microsoft Research
St. George House
1 Guildhall Street
Cambridge, CB2 3NH, UK
cmbishop@microsoft.com

## Abstract

Gaussian processes provide natural non-parametric prior distributions over regression functions. In this paper we consider regression problems where there is noise on the output, and the variance of the noise depends on the inputs. If we assume that the noise is a smooth function of the inputs, then it is natural to model the noise variance using a second Gaussian process, in addition to the Gaussian process governing the noise-free output value. We show that prior uncertainty about the parameters controlling both processes can be handled and that the posterior distribution of the noise rate can be sampled from using Markov chain Monte Carlo methods. Our results on a synthetic data set give a posterior noise variance that well-approximates the true variance.

## 1 Background and Motivation

A very natural approach to regression problems is to place a prior on the kinds of function that we expect, and then after observing the data to obtain a posterior. The prior can be obtained by placing prior distributions on the weights in a neural

network, although we would argue that it is perhaps more natural to place priors directly over functions. One tractable way of doing this is to create a *Gaussian process prior*. This has the advantage that predictions can be made from the posterior using only matrix multiplication for fixed hyperparameters and a global noise level. In contrast, for neural networks (with fixed hyperparameters and a global noise level) it is necessary to use approximations or Markov chain Monte Carlo (MCMC) methods. Rasmussen (1996) has demonstrated that predictions obtained with Gaussian processes are as good as or better than other state-of-the art predictors.

In much of the work on regression problems in the statistical and neural networks literatures, it is assumed that there is a global noise level, independent of the input vector $x$. The book by Bishop (1995) and the papers by Bishop (1994), MacKay (1995) and Bishop and Qazaz (1997) have examined the case of input-dependent noise for parametric models such as neural networks. (Such models are said to be *heteroscedastic* in the statistics literature.) In this paper we develop the treatment of an input-dependent noise model for Gaussian process regression, where the noise is assumed to be Gaussian but its variance depends on $x$. As the noise level is non-negative we place a Gaussian process prior on the log noise level. Thus there are two Gaussian processes involved in making predictions: the usual Gaussian process for predicting the function values (the $y$-process), and another one (the $z$-process) for predicting the log noise level. Below we present a Markov chain Monte Carlo method for carrying out inference with this model and demonstrate its performance on a test problem.

## 1.1 Gaussian processes

A stochastic process is a collection of random variables $\{Y(x)|x \in X\}$ indexed by a set $X$. Often $X$ will be a space such as $\mathcal{R}^d$ for some dimension $d$, although it could be more general. The stochastic process is specified by giving the probability distribution for every finite subset of variables $Y(x_1), \ldots, Y(x_k)$ in a consistent manner. A Gaussian process is a stochastic process which can be fully specified by its mean function $\mu(x) = E[Y(x)]$ and its covariance function $C_P(x, x') = E[(Y(x) - \mu(x))(Y(x') - \mu(x'))]$; any finite set of points will have a joint multivariate Gaussian distribution. Below we consider Gaussian processes which have $\mu(x) \equiv 0$. This assumes that any known offset or trend in the data has been removed. A non-zero $\mu(x)$ is easily incorporated into the framework at the expense of extra notational complexity.

A *covariance function* is used to define a Gaussian process; it is a parametrised function from pairs of $x$-values to their covariance. The form of the covariance function that we shall use for the prior over functions is given by

$$C_Y(x^{(i)}, x^{(j)}) = v_Y \exp\left(-\frac{1}{2}\sum_{l=1}^{d} w_{Yl}(x_l^{(i)} - x_l^{(j)})^2\right) + J_Y \delta(i, j) \qquad (1)$$

where $v_Y$ specifies the overall $y$-scale and $w_{Yl}^{-1/2}$ is the length-scale associated with the $l$th coordinate. $J_Y$ is a "jitter" term (as discussed by Neal, 1997), which is added to prevent ill-conditioning of the covariance matrix of the outputs. $J_Y$ is a typically given a small value, e.g. $10^{-6}$.

For the prediction problem we are given $n$ data points $\mathcal{D} = ((x_1, t_1), (x_2, t_2),$

$\ldots, (\boldsymbol{x}_n, t_n))$, where $t_i$ is the observed output value at $\boldsymbol{x}_i$. The $t$'s are assumed to have been generated from the true $y$-values by adding independent Gaussian noise whose variance is $\boldsymbol{x}$-dependent. Let the noise variance at the $n$ data points be $\boldsymbol{r} = (r(\boldsymbol{x}_1), r(\boldsymbol{x}_2), \ldots, r(\boldsymbol{x}_n))$. Given the assumption of a Gaussian process prior over functions, it is a standard result (e.g. Whittle, 1963) that the predictive distribution $P(t^*|\boldsymbol{x}^*)$ corresponding to a new input $\boldsymbol{x}^*$ is $t^* \sim N(\hat{t}(\boldsymbol{x}^*), \sigma^2(\boldsymbol{x}^*))$, where

$$\hat{t}(\boldsymbol{x}^*) = \boldsymbol{k}_Y^T(\boldsymbol{x}^*)(K_Y + K_N)^{-1}\boldsymbol{t} \tag{2}$$

$$\sigma^2(\boldsymbol{x}^*) = C_Y(\boldsymbol{x}^*, \boldsymbol{x}^*) + r(\boldsymbol{x}^*) - \boldsymbol{k}_Y^T(\boldsymbol{x}^*)(K_Y + K_N)^{-1}\boldsymbol{k}_Y(\boldsymbol{x}^*) \tag{3}$$

where the noise-free covariance matrix $K_Y$ satisfies $[K_Y]_{ij} = C_Y(\boldsymbol{x}_i, \boldsymbol{x}_j)$, and $\boldsymbol{k}_Y(\boldsymbol{x}^*) = (C_Y(\boldsymbol{x}^*, \boldsymbol{x}_1), \ldots, C_Y(\boldsymbol{x}^*, \boldsymbol{x}_n))^T$, $K_N = \text{diag}(\boldsymbol{r})$ and $\boldsymbol{t} = (t_1, \ldots, t_n)^T$, and $\sqrt{\sigma^2(\boldsymbol{x}^*)}$ gives the "error bars" or confidence interval of the prediction.

In this paper we do not specify a functional form for the noise level $r(\boldsymbol{x})$ but we do place a prior over it. An independent Gaussian process (the $z$-process) is defined to be the log of the noise level. Its values at the training data points are denoted by $\boldsymbol{z} = (z_1, \ldots, z_n)$, so that $\boldsymbol{r} = (\exp(z_1), \ldots, \exp(z_n))$. The prior for $z$ has a covariance function $C_Z(\boldsymbol{x}^{(i)}, \boldsymbol{x}^{(j)})$ similar to that given in equation 1, although the parameters $v_Z$ and the $w_{Zl}$'s can be chosen to be different to those for the $y$-process. We also add the jitter term $J_Z\delta(i, j)$ to the covariance function for $Z$, where $J_Z$ is given the value $10^{-2}$. This value is larger than usual, for technical reasons discussed later.

We use a zero-mean process for $z$ which carries a prior assumption that the average noise rate is approximately 1 (being $e$ to the power of components of $\boldsymbol{z}$). This is suitable for the experiment described in section 3. In general it is easy to add an offset to the $z$-process to shift the prior noise rate.

## 2 An input-dependent noise process

We discuss, in turn, sampling the noise rates and making predictions with fixed values of the parameters that control both processes, and sampling from the posterior on these parameters.

### 2.1 Sampling the Noise Rates

The predictive distribution for $t^*$, the output at a point $\boldsymbol{x}^*$, is $P(t^*|\boldsymbol{t}) = \int P(t^*|\boldsymbol{t}, \boldsymbol{r}(\boldsymbol{z}))P(\boldsymbol{z}|\boldsymbol{t})d\boldsymbol{z}$. Given a $z$ vector, the prediction $P(t^*|\boldsymbol{t}, \boldsymbol{r}(\boldsymbol{z}))$ is Gaussian with mean and variance given by equations 2 and 3, but $P(\boldsymbol{z}|\boldsymbol{t})$ is difficult to handle analytically, so we use a Monte Carlo approximation to the integral. Given a representative sample $\{\boldsymbol{z}_1, \ldots, \boldsymbol{z}_k\}$ of log noise rate vectors we can approximate the integral by the sum $\frac{1}{k}\sum_j P(t^*|\boldsymbol{t}, \boldsymbol{r}(\boldsymbol{z}_j))$.

We wish to sample from the distribution $P(\boldsymbol{z}|\boldsymbol{t})$. As this is quite difficult, we sample instead from $P(\boldsymbol{y}, \boldsymbol{z}|\boldsymbol{t})$; a sample for $P(\boldsymbol{z}|\boldsymbol{t})$ can then be obtained by ignoring the $y$ values. This is a similar approach to that taken by Neal (1997) in the case of Gaussian processes used for classification or robust regression with $t$-distributed noise. We find that

$$P(\boldsymbol{y}, \boldsymbol{z}|\boldsymbol{t}) \propto P(\boldsymbol{t}|\boldsymbol{y}, \boldsymbol{r}(\boldsymbol{z}))P(\boldsymbol{y})P(\boldsymbol{z}). \tag{4}$$

We use Gibbs sampling to sample from $P(\boldsymbol{y}, \boldsymbol{z}|\boldsymbol{t})$ by alternately sampling from $P(\boldsymbol{z}|\boldsymbol{y}, \boldsymbol{t})$ and $P(\boldsymbol{y}|\boldsymbol{z}, \boldsymbol{t})$. Intuitively were are alternating the "fitting" of the curve (or

$y$-process) with "fitting" the noise level ($z$-process). These two steps are discussed in turn.

- *Sampling from $P(y|t, z)$*

For $y$ we have that

$$P(y|t, z) \propto P(t|y, r(z))P(y) \tag{5}$$

where

$$P(t|y, r(z)) = \prod_{i=1}^{n} \frac{1}{(2\pi r_i)^{1/2}} \exp\left(-\frac{(t_i - y_i)^2}{2r_i}\right). \tag{6}$$

Equation (6) can also be written as $P(t|y, r(z)) \sim N(t, K_N)$. Thus $P(y|t, z)$ is a multivariate Gaussian with mean $(K_Y^{-1} + K_N^{-1})^{-1} K_N^{-1} t$ and covariance matrix $(K_Y^{-1} + K_N^{-1})^{-1}$ which can be sampled by standard methods.

- *Sampling from $P(z|t, y)$*

For fixed $y$ and $t$ we obtain

$$P(z|y, t) \propto P(t|y, z)P(z). \tag{7}$$

The form of equation 6 means that it is not easy to sample $z$ as a vector. Instead we can sample its components separately, which is a standard Gibbs sampling algorithm. Let $z_i$ denote the $i$th component of $z$ and let $z_{-i}$ denote the remaining components. Then

$$P(z_i|z_{-i}, y, t) \propto \frac{1}{(2\pi \exp(z_i))^{1/2}} \exp\left(-\frac{(t_i - y_i)^2}{2\exp(z_i)}\right) P(z_i|z_{-i}). \tag{8}$$

$P(z_i|z_{-i})$ is the distribution of $z_i$ conditioned on the values of $z_{-i}$. The computation of $P(z_i|z_{-i})$ is very similar to that described by equations (2) and (3), except that $C_Y(\cdot, \cdot)$ is replaced by $C_Z(\cdot, \cdot)$ and there is no noise so that $r(\cdot)$ will be identically zero.

We sample from $P(z_i|z_{-i}, y, t)$ using rejection sampling. We first sample from $P(z_i|z_{-i})$, and then reject according to the term $\exp\{-z_i/2 - \frac{1}{2}(t_i - y_i)^2 \exp(-z_i)\}$ (the likelihood of local noise rate $z_i$), which can be rescaled to have a maximum value of 1 over $z_i$. Note that it is not necessary to perform a separate matrix inversion for each $i$ when computing the $P(z_i|z_{-i})$ terms; the required matrices can be computed efficiently from the inverse of $K_Z$. We find that the average rejection rate is approximately two-thirds, which makes the method we currently use reasonably efficient. Note that it is also possible to incorporate the term $\exp(-z_i/2)$ from the likelihood into the mean of the Gaussian $P(z_i|z_{-i})$ to reduce the rejection rate.

As an alternative approach, it is possible to carry out Gibbs sampling for $P(z_i|z_{-i}, t)$ without explicitly representing $y$, using the fact that $\log P(t|z) = -\frac{1}{2}\log |K| - \frac{1}{2}t^T K^{-1} t + const$, where $K = K_Y + K_N$. We have implemented this and found similar results to those obtained using sampling of the $y$'s. However, explicitly representing the $y$-process is useful when adapting the parameters, as described in section 2.3.

## 2.2   Making predictions

So far we have explained how to obtain a sample from $P(z|t)$. To make predictions we use

$$P(t^*|t) \simeq \frac{1}{k} \sum_j P(t^*|t, r(z_j)). \tag{9}$$

However, $P(t^*|t, r(z_j))$ is not immediately available, as $z^*$, the noise level at $x^*$ is unknown. In fact

$$P(t^*|t, r(z_j)) = \int P(t^*|z^*, t, r(z_j)) P(z^*|z_j, t) \, dz^*. \tag{10}$$

$P(z^*|z_j, t)$ is simply a Gaussian distribution for $z^*$ conditioned on $z_j$, and is obtained in a similar way to $P(z_i|z_{-i})$. As $P(t^*|z^*, t, r(z_j))$ is a Gaussian distribution as given by equations (2) and (3), $P(t^*|t, r(z_j))$ is an infinite mixture of Gaussians with weights $P(z^*|z_j)$. Note, however, that each of these components has the same mean $\hat{t}(x^*)$ as given by equation (2), but a different variance.

We approximate $P(t^*|t, r(z_j))$ by taking $s = 10$ samples of $P(z^*|z_j)$ and thus obtain a mixture of $s$ Gaussians as the approximating distribution. The approximation for $P(t^*|t)$ is then obtained by averaging these $s$-component mixtures over the $k$ samples $z_1, \ldots, z_k$ to obtain an $sk$-component mixture of Gaussians.

## 2.3   Adapting the parameters

Above we have described how to obtain a sample from the posterior distribution $P(z|t)$ and to use this to make predictions, based on the assumption that the parameters $\theta_Y$ (i.e. $v_Y, J_Y, w_{Y1}, \ldots, w_{Yd}$) and $\theta_Z$ (i.e. $v_Z, J_Z, w_{Z1}, \ldots, w_{Zd}$) have been set to the correct values. In practice we are unlikely to know what these settings should be, and so introduce a hierarchical model, as shown in Figure 1. This graphical model shows that the joint probability distribution decomposes as $P(\theta_Y, \theta_Z, y, z, t) = P(\theta_Y)P(\theta_Z)P(y|\theta_Y)P(z|\theta_Z)P(t|y, z)$.

Our goal now becomes to obtain a sample from the posterior $P(\theta_Y, \theta_Z, y, z|t)$, which can be used for making predictions as before. (Again, the $y$ samples are not needed for making predictions, but they will turn out to be useful for sampling $\theta_Y$.) Sampling from the joint posterior can be achieved by interleaving updates of $\theta_Y$ and $\theta_Z$ with $y$ and $z$ updates. Gibbs sampling for $\theta_Y$ and $\theta_Z$ is not feasible as these parameters are buried deeply in the $K_Y$ and $K_N$ matrices, so we use the Metropolis algorithm for their updates. As usual, we consider moving from our current state $\theta = (\theta_Y, \theta_Z)$ to a new state $\bar{\theta}$ using a proposal distribution $J(\theta, \bar{\theta})$. In practice we take $J$ to be an isotropic Gaussian centered on $\theta$. Denote the ratio of $P(\theta_Y)P(\theta_Z)P(y|\theta_Y)P(z|\theta_Z)$ in states $\bar{\theta}$ and $\theta$ by $r$. Then the proposed state $\bar{\theta}$ is accepted with probability $\min\{r, 1\}$.

It would also be possible to use more sophisticated MCMC algorithms such as the Hybrid Monte Carlo algorithm which uses derivative information, as discussed in Neal (1997).

## 3   Results

We have tested the method on a one-dimensional synthetic problem. 60 data points

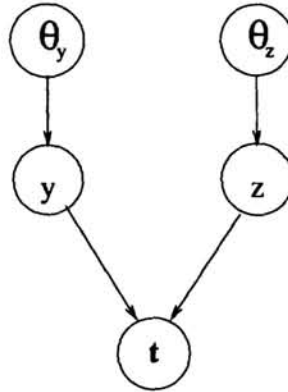

Figure 1: The hierarchical model including parameters.

were generated from the function $y = 2\sin(2\pi x)$ on $[0, 1]$ by adding independent Gaussian noise. This noise has a standard deviation that increases linearly from 0.5 at $x = 0$ to 1.5 at $x = 1$. The function and the training data set are illustrated in Figure 2(a).

As the parameters are non-negative quantities, we actually compute with their log values. $\log v_Y, \log v_Z, \log w_Y$ and $\log w_Z$ were given $N(0, 1)$ prior distributions. The jitter values were fixed at $J_Y = 10^{-6}$ and $J_Z = 10^{-2}$. The relatively large value for $J_Z$ assists the convergence of the Gibbs sampling, since it is responsible for most of the variance of the conditional distribution $P(z_i|z_{-i})$. The broadening of this distribution leads to samples whose likelihoods are more variable, allowing the likelihood term (used for rejection) to be more influential.

In our simulations, on each iteration we made three Metropolis updates for the parameters, along with sampling from all of the $y$ and $z$ variables. The Metropolis proposal distribution was an isotropic Gaussian with variance 0.01. We ran for 3000 iterations, and discarded the first one-third of iterations as "burn-in", after which plots of each of the parameters seemed to have settled down. The parameters and $z$ values were stored every 100 iterations. In Figure 2(b) the average standard deviation of the inferred noise has been plotted, along with with two standard deviation error-bars. Notice how the standard deviation increases from left to right, in close agreement with the data generator.

Studying the posterior distributions of the parameters, we find that the $y$-lengthscale $\lambda_Y \overset{def}{=} (w_Y)^{-1/2}$ is well localized around $0.22 \pm 0.1$, in good agreement with the wavelength of the sinusoidal generator. (For the covariance function in equation 1, the expected number of zero crossings per unit length is $1/\pi\lambda_Y$.) $(w_Z)^{-1/2}$ is less tightly constrained, which makes sense as it corresponds to a longer wavelength process, and with only a short segment of data available there is still considerable posterior uncertainty.

## 4   Conclusions

We have introduced a natural non-parametric prior on variable noise rates, and given an effective method of sampling the posterior distribution, using a MCMC

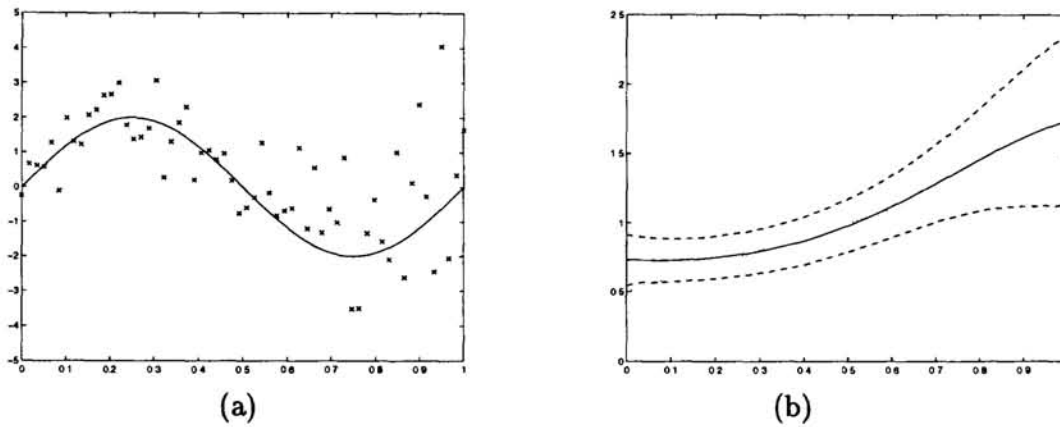

(a)                                    (b)

Figure 2: (a) shows the training set (crosses); the solid line depicts the $x$-dependent mean of the output. (b) The solid curve shows the average standard deviation of the noise process, with two standard deviation error bars plotted as dashed lines. The dotted line indicates the true standard deviation of the data generator.

method. When applied to a data set with varying noise, the posterior noise rates obtained are well-matched to the known structure. We are currently experimenting with the method on some more challenging real-world problems.

## Acknowledgements

This work was carried out at Aston University under EPSRC Grant Ref. GR/K 51792 *Validation and Verification of Neural Network Systems.*

## References

[1] C.M. Bishop (1994). Mixture Density Networks. Technical report NCRG/94/001, Neural Computing Research Group, Aston University, Birmingham, UK.

[2] C.M. Bishop (1995). *Neural Networks for Pattern Recognition.* Oxford University Press.

[3] C.M. Bishop and C. Qazaz (1997). Regression with Input-dependent Noise: A Bayesian Treatment. In M. C. Mozer, M. I. Jordan and T. Petsche (Eds) *Advances in Neural Information Processing Systems 9* Cambridge MA MIT Press.

[4] D. J. C. MacKay (1995). Probabilistic networks: new models and new methods. In F. Fogelman-Soulie and P. Gallinari (Eds), *Proceedings ICANN'95 International Conference on Neural Networks*, pp. 331-337. Paris, EC2 & Cie.

[5] R. Neal (1997). Monte Carlo Implementation of Gaussian Process Models for Bayesian Regression and Classification. Technical Report 9702, Department of Statistics, University of Toronto. Available from http://www.cs.toronto.edu/~radford/.

[6] C.E. Rasmussen (1996). *Evaluation of Gaussian Processes and Other Methods for Nonlinear Regression.* PhD thesis, Department of Computer Science, University of Toronto. Available from http://www.cs.utoronto.ca/~carl/.

[7] C.K.I. Williams and C.E. Rasmussen (1996). Gaussian Processes for Regression. In D. S. Touretzky, M. C. Mozer and M. E. Hasselmo *Advances in Neural Information Processing Systems 8* pp. 514-520, Cambridge MA MIT Press.

[8] P. Whittle (1963). *Prediction and regulation by linear least-square methods.* English Universities Press.